# On the accuracy of $\ell_1$-filtering of signals with block-sparse structure

**Anatoli Juditsky**[*]     Fatma Kılınç Karzan[†]     Arkadi Nemirovski[‡]     Boris Polyak[§]

## Abstract

We discuss new methods for the recovery of signals with block-sparse structure, based on $\ell_1$-minimization. Our emphasis is on the efficiently computable error bounds for the recovery routines. We optimize these bounds with respect to the method parameters to construct the estimators with improved statistical properties. We justify the proposed approach with an oracle inequality which links the properties of the recovery algorithms and the best estimation performance.

## 1 Introduction

Suppose an observation $y \in \mathbb{R}^m$ is available where

$$y = Ax + u + D\xi. \tag{1}$$

Here $A$ is a given $m \times n$ sensing matrix, $x \in \mathbb{R}^n$ is an unknown vector, $u$ is an unknown (deterministic) *nuisance* parameter, known to belong to a certain set $\mathcal{U} \subset \mathbb{R}^m$, $D \in \mathbb{R}^{m \times m}$ is known noise intensity matrix, and $\xi \in \mathbb{R}^m$ is random noise with standard normal distribution.

We aim to recover a linear transformation $w = Bx$ of the signal x, where $B$ is a given $N \times n$ matrix, under the assumption that $w$ is *block-sparse*. Namely, the space $\mathcal{W}(= \mathbb{R}^N)$ where $w$ "lives" is represented as $\mathcal{W} = \mathbb{R}^{n_1} \times ... \times \mathbb{R}^{n_K}$, so that $w = Bx \in \mathbb{R}^N$ is a block vector: $w = [w[1]; ...; w[K]]$ with blocks $w[k] = B[k]x \in \mathbb{R}^{n_k}$, $1 \le k \le K$, where $B[k]$, $1 \le k \le K$ are $n_k \times n$ matrices. The $s$-block-sparsity of $w$ means that at most a given number $s$ of the blocks $u[k]$, $1 \le k \le K$, are nonzero.

To motivate the interest for the presented model, let us consider two examples.

**Tracking of a singularly perturbed linear system**   Consider a discrete-time linear system

$$z[i] = Gz[i-1] + w[i] + F\eta[i], \quad i = 1, 2, ..., \; z[0] \in \mathbb{R}^d,$$

where $F\eta[i]$ are random perturbations with $\eta[i]$ being i.i.d. standard normal vectors $\eta[i] \sim \mathcal{N}(0, I_d)$, and $G$, $F \in \mathbb{R}^{d \times d}$ are known matrices. We assume that the perturbation vectors $w[i] \in \mathbb{R}^d$ $i = 1, 2, ...$ are mostly zeros, but a small proportion of $w[i]$ are nonvanishing unknown vectors in $\mathbb{R}^d$. Suppose that we are given the linear noisy observation $y \in \mathbb{R}^m$, such that $y = A[z[0]; ...z[K]] + \sigma\xi$, where the matrix $A \in \mathbb{R}^{m \times d(K+1)}$ and the noise intensity $\sigma > 0$ are known, and $\xi \sim \mathcal{N}(0, I_m)$. Given $y$, our objective is to recover the sequence of perturbations $w = [w[k]]_{k=1}^K$, $w[k] \in \mathbb{R}^d$, and the trajectory $z$ of the system.

---

[*]LJK, Université J. Fourier, B.P. 53, 38041 Grenoble Cedex 9, France, `Anatoli.Juditsky@imag.fr`

[†]Carnegie Mellon University, Pittsburgh, Pennsylvania 15213, USA, `fkilinc@andrew.cmu.edu`

[‡]Georgia Institute of Technology, Atlanta, Georgia 30332, USA, `nemirovs@isye.gatech.edu`

Research of the second and the third authors was supported by the Office of Naval Research grant #240667V.

[§]Institute of Control Sciences of Russian Academy of Sciences, Moscow 117997, Russia, `boris@ipu.rssi.ru`

To fit the tracking problem into the basic framework let us decompose $z = x + \zeta$, where $x = [x[0]; ...; x[K]]$ with $x[i] = Gx[i-1] + w[i]$, $x[0] = z[1]$, and $\zeta = [\zeta[0]; ...; \zeta[K]]$ with $\zeta[i] = G\zeta[i-1] + F\eta[i]$, $\zeta[0] = 0$. Then

$$y = Ax + [A\zeta + \sigma\xi],$$

where the distribution of $A\zeta + \sigma\xi$ is normal with zero mean and the covariance matrix $D^2 = AVA^T + \sigma^2 I$. Here the $Kd \times Kd$ covariance matrix $V$ of $\zeta$ has the block structure with blocks

$$V^{k,\ell} = \text{Cov}(\zeta[k], \zeta[\ell]) = \sum_{i=1}^{\ell \wedge k} G^{\ell-i} FF^T (G^T)^{k-i},$$

with $\sum_{1}^{0} = 0$, by convention.

**Image reconstruction with regularization by Total Variation (TV)** [21, 7] Here one looks to recover an image $Z \in \mathbb{R}^{n_1 \times n_2}$ from a blurred noisy observation $y$: $y = Az + \sigma\xi$, $y \in \mathbb{R}^m$, where $z = \text{Col}(Z) \in \mathbb{R}^n$, $n = n_1 n_2$, $A \in \mathbb{R}^{m \times n}$ is the matrix of discrete convolution, $\sigma > 0$ is known, and $\xi \sim \mathcal{N}(0, I_m)$. We assume that the image $z$ may be decomposed as $z = x + v$, where $v$ is a "regular component", which is modeled by restricting $v$ to belong to the set $\mathcal{V}$ of "smooth images"; let $w = Bx \in \mathbb{R}^{n \times 2}$, be the (discretized) gradient of the $x$-component at the points of the grid. In this example $Bx$ naturally splits into 2-dimensional blocks and TV is nothing but the sum of $\ell_2$ norms of these blocks. We suppose that $w$ is (nearly) sparse. When denoting $u = Av$ we come to the observation model $y = Ax + u + D\xi$, with $u \in \mathcal{U} = A\mathcal{V}$, $D = \sigma I$, and $\xi \sim \mathcal{N}(0, I_m)$.

The recovery routines we consider are based on the *block-$\ell_1$ minimization*, i.e., the estimate $\widehat{w}(y)$ of $w = Bx$ is $\widehat{w} = B\widehat{x}(y)$, where $\widehat{x}(y)$ is obtained by minimizing the norm $\sum_{k=1}^{K} \|B[k]z\|_{(k)}$ over signals $z \in \mathbb{R}^n$ with $Az$ "fitting," in certain precise sense, the observations $y$. Above, $\|\cdot\|_{(k)}$ are given in advance norms on the spaces $\mathbb{R}^{n_k}$ where the blocks of $w$ take their values.

In the sequel we refer to the given in advance collection $\mathcal{B} = (B, n_1, ..., n_K, \|\cdot\|_{(1)}, ..., \|\cdot\|_{(K)})$ as to the *representation structure*. Given such a structure $\mathcal{B}$ and matrix $A$, our goal is to understand how well one can recover the $s$-block-sparse transform $Bx$ by appropriately implemented block-$\ell_1$ minimization.

**Related Compressed Sensing research** Our situation and goal form a straightforward extension of the usual block sparsity Compressed Sensing framework. Indeed, the *standard representation structure* $B = I_n, n_k = 1, \|\cdot\|_{(k)} = |\cdot|, 1 \le k \le K = n$, leads to the standard Compressed Sensing setting – recovering a sparse signal $x \in \mathbb{R}^n$ from its noisy observations (1) via $\ell_1$ minimization. With the same $B = I_n$ and nontrivial block structure $\{n_k, \|\cdot\|_{(k)}\}_{k=1}^{K}$, we arrive at *block-sparsity* and related block-$\ell_1$ minimization routines considered in numerous recent papers. There is a number of applications where block-sparsity seem to arise naturally (see, e.g., [10] and references therein). Several methods of estimation and selection extending the plain $\ell_1$-minimization to block sparsity were proposed and investigated recently. Most of the related research is focused so far on *block regularization schemes* – Lasso-type algorithms

$$\widehat{x}(y) \in \underset{z=[z[1];...;z[K]] \in \mathbb{R}^n = \mathbb{R}^{n_1} \times ... \times \mathbb{R}^{n_K}}{\text{Argmin}} \left\{ \|Az - y\|_2^2 + \lambda L_{[1,2]}(z) \right\}, \quad L_{[1,2]}(z) = \sum_{k=1}^{K} \|z[k]\|_2,$$

$\|\cdot\|_2$ being the "usual" $\ell_2$-norm on $\mathbb{R}^{n_k}$. In particular, the huge literature on plain Lasso has a significant counterpart on group Lasso, see, e.g., [1, 2, 8, 9, 10, 11, 16, 19, 20, 22, 23], and references therein. Another classical Compressed Sensing estimator, Dantzig Selector, is studied in the block-sparse case in [12, 17]. The available theoretical results allows to bound the errors of recovery in terms of magnitude of the observation noise and "$s$-concentration" of the true signal $x$ (that is, its $L_{[1,2]}$ distance from the space of signals with at most $s$ nonzero blocks). Typically, these results deal with the quadratic risks of estimation and rely on a natural block analogy ("Block RIP," see, e.g., [10]) of the celebrated Restricted Isometry property for the sensing matrix $A$, introduced by Candés and Tao [5], or on a block analogy [18] of the Restricted Eigenvalue property from [3].

**Contributions of this work** To the best of our knowledge, the conditions used when studying theoretical properties of block-sparse recovery (with a notable exception of the Mutual Block In-coherence condition of [9]) are unverifiable. The latter means that given the matrix $A$, one cannot

answer in any reasonable time if the (block-) RI or RE property holds with given parameters. While the efficient verifiability of a condition is by no means necessary for a condition to be meaningful and useful, we believe also that verifiability has its value and is worthy of being investigated. In particular, the verifiability property allows us to design new recovery routines with explicit confidence bounds for the recovery error and optimize these bounds with respect to the method parameters. Thus, the major novelty in what follows is the emphasis on *verifiable* conditions on $A$ and the representation structure which *guarantee* good recovery of $Bx$ from noisy observations of $Ax$, provided that $Bx$ is nearly $s$-block-sparse, and the observation noise is low. In this respect, this work extends the results of [15, 13, 14], where $\ell_1$-recovery of the "usual" sparse vectors was considered (in the first two papers – in the case of uncertain-but-bounded observation errors, and in the third – in the case of Gaussian observation noise). We propose new routines of block-sparse recovery which explicitly utilize the *verifiability certificate – the contrast matrix*, and show how these routines may be tuned to attain the best performance bounds.

The rest of the manuscript is organized as follows. In Section 2 we give the detailed problem statement and introduce the family $\mathbf{Q}_{s,q}$, $1 \leq q \leq \infty$, of conditions which underly the subsequent developments. Then in Section 2.3 we introduce the recovery routines and provide the bounds for their risks. We discuss the properties of conditions $\mathbf{Q}_{s,q}$ in Section 3. In particular, in Section 3.1 we show how one can efficiently verify (the strongest from the family $\mathbf{Q}_{s,q}$) condition $\mathbf{Q}_{s,\infty}$. Then in Section 3.2 we provide an oracle inequality which shows that the condition $\mathbf{Q}_{s,\infty}$ is also necessary for recovery of block-sparse signals in $\ell_\infty$-norm.

# 2 Accuracy bounds for $\ell_1$-recovery routines

## 2.1 Problem statement and notations

Let $w = Bx \in \mathcal{W} = \mathbb{R}^{n_1} \times \ldots \times \mathbb{R}^{n_K}$. To streamline the presentation, we restrict ourselves to the case where all the norms $\|\cdot\|_{(k)}$ on the factors of the representations are the usual $\ell_r$-norms, $1 \leq r \leq \infty$: $\|w[k]\|_r = \left(\sum_{i=1}^{n_k} |w[k]_i|^r\right)^{\frac{1}{r}}$, i.e., the representation structures we consider are $\mathcal{B} = (B, n_1, \ldots, n_K, \|\cdot\|_r)$. Let $r_* = \frac{r}{r-1}$, so that $\|\cdot\|_{r_*}$ is the norm conjugate to $\|\cdot\|_r$. A vector $w = [w[1]; \ldots; w[K]]$ from $\mathcal{W}$ is called $s$-*block-sparse*, if the number of nonzero blocks $w[k] \in \mathbb{R}^{n_k}$ in $w$ is at most $s$.

For $w \in \mathcal{W}$, we call the number $\|w[k]\|_r$ the *magnitude* of $k$-th block in $w$, and denote by $w^s$ the representation vector obtained from $w$ by zeroing out all but the $s$ largest in magnitude blocks in $w$ (with the ties resolved arbitrarily). For $w \in \mathcal{W}$ and $1 \leq p \leq \infty$, we denote by $L_{[p,r]}(w)$ the $\|\cdot\|_p$-norm of the vector $[\|w[1]\|_r; \ldots; \|w[K]\|_r]$, so that $L_{[p,r]}(\cdot)$ is a norm on $\mathcal{W}$ with the conjugate norm $L_{[p,r]}^*(w) = \|[\|w[1]\|_{(r_*)}; \ldots; \|w[K]\|_{(r_*)}]\|_{p_*}$, $p_* = \frac{p}{p-1}$. Given a positive integer $s \leq K$, we set $L_{s,[p,r]}(w) = L_{[p,r]}(w^s)$; note that $L_{s,[p,r]}(\cdot)$ is a norm on $\mathcal{W}$. When the representation structure $\mathcal{B}$ of $x$ (and thus the norm $\|\cdot\|_r$) is fixed, we use the notation $L_p$, $L_p^*$, and $L_{s,p}$, instead of $L_{[p,r]}$, $L_{[p,r]}^*$, and $L_{s,[p,r]}$.

The recovery problem we are interested in is as follows: suppose we are given an indirect observation (cf (1))

$$y = Ax + u + D\xi$$

of unknown signal $x \in \mathbb{R}^n$. Here $A \in \mathbb{R}^{m \times n}$, $u + D\xi$ is the observation error; in this error, $u$ is an unknown *nuisance* known to belong to a given compact convex set $\mathcal{U} \subset \mathbb{R}^m$ symmetric w.r.t. the origin, $D \in \mathbb{R}^{m \times m}$ is known, and $\xi \sim \mathcal{N}(0, I_m)$.

We want to recover $x$ and the representation $w = Bx$ of $x$, knowing in advance that this representation is *nearly $s$-block-sparse*, for some given $s$. Specifically, we consider the set

$$X(s, \upsilon) = \{x \in \mathbb{R}^n : L_1(Bx - [Bx]^s) \leq \upsilon\}.$$

A recovery routine is a Borel function $\widehat{x}(y) : \mathbb{R}^m \to \mathbb{R}^n$ and we characterize the performance of such a routine by its $L_p$-risk of recovery $\widehat{w}(y) = B\widehat{x}(y)$ of $w = Bx$:

$$\mathrm{Risk}_p(\widehat{w}(\cdot)|s, D, \upsilon, \epsilon)$$
$$= \inf \left\{ \delta : \mathrm{Prob}_\xi \left\{ L_p\left(\widehat{w}(y) - w\right) \leq \delta \; \forall (u \in \mathcal{U}, \, x \in X(s, \upsilon)) \right\} \geq 1 - \epsilon \right\}.$$

here $0 \le \epsilon \le 1$ and $1 \le p \le \infty$. In other words, $\text{Risk}_p(\widehat{w}(\cdot)|s, D, \upsilon, \epsilon) \le \delta$ if and only if there exists a set $\Xi \in \mathbb{R}^m$ of "good realizations of $\xi$" such that $\text{Prob}\{\xi \in \Xi\} \ge 1 - \epsilon$ and the $L_{[p,r]}$-norm of $B[\widehat{x}(y) - x]$ is $\le \delta$ whenever $\xi \in \Xi$, $u \in \mathcal{U}$, and whenever $x \in \mathbb{R}^n$ is such that $Bx$ can be approximated by $s$-block-sparse representation vector within accuracy $\upsilon$ (measured in the $L_{[1,r]}$-norm).

## 2.2   Condition $\mathbf{Q}_{s,q}(\kappa)$

Let a sensing matrix $A$ and a representation structure $\mathcal{B} = (B, n_1, ..., n_K, \|\cdot\|_r)$ be given, and let $s \le K$ be a positive integer, $q \in [1, \infty]$ and $\kappa > 0$. We say that a pair $(H, \|\cdot\|)$, where $H \in \mathbb{R}^{m \times M}$ and $\|\cdot\|$ is a norm on $\mathbb{R}^M$, satisfies the condition $\mathbf{Q}_{s,q}(\kappa)$ associated with the matrix $A$ and $\mathcal{B}$, if

$$\forall x \in \mathbb{R}^n : \ L_{s,q}(Bx) \le s^{\frac{1}{q}}\|H^T Ax\| + \kappa s^{\frac{1}{q}-1}L_1(Bx). \tag{2}$$

The following is an evident observation.

**Observation 2.1** *Given $A$ and a representation structure $\mathcal{B}$, let $(H, \|\cdot\|)$ satisfy $\mathbf{Q}_{s,q}(\kappa)$. Then $(H, \|\cdot\|)$ satisfies $\mathbf{Q}_{s,q'}(\kappa')$ for all $q' \in (1, q)$ and $\kappa' \ge \kappa$. Besides this, if $s' \le s$ is a positive integer, $((s/s')^{\frac{1}{q}} H, \|\cdot\|)$ satisfies $\mathbf{Q}_{s',q}((s'/s)^{1-\frac{1}{q}}\kappa)$.*

Whenever $(B, n_1, ..., n_K, \|\cdot\|_r)$ is the standard representation structure, meaning that $B$ is the identity matrix, $n_1 = 1$, and $\|\cdot\|_r = |\cdot|$, the condition $\mathbf{Q}_{s,q}(\kappa)$ reduces to the condition $\mathbf{H}_{s,q}(\kappa)$ introduced in [14].

## 2.3   $\ell_1$-Recovery Routines

We consider two block-sparse recovery routines.

**Regular $\ell_1$ recovery**   is given by

$$\widehat{x}_{\text{reg}}(y) \in \underset{z}{\text{Argmin}} \ \left\{ L_1(Bz) : \|H^T(Az - y)\| \le \rho \right\},$$

where $H \in \mathbb{R}^{m \times M}$, $\|\cdot\|$ and $\rho > 0$ are parameters of the construction.

**Theorem 2.1** *Let $s$ be a positive integer, $q \in [1, \infty]$ and $\kappa \in (0, 1/2)$. Let also $\epsilon \in (0, 1)$. Assume that the parameters $H$, $\|\cdot\|$, $\rho$ of the regular $\ell_1$-recovery are such that*

   **A**. *$(H, \|\cdot\|)$ satisfies the condition $\mathbf{Q}_{s,q}(\kappa)$ associated with the matrix $A$ and the representation structure $\mathcal{B}$;*

   **B**. *There exists a set $\Xi \subset \mathbb{R}^m$ such that $\text{Prob}(\xi \in \Xi) \ge 1 - \epsilon$ and*

$$\|H^T(u + D\xi)\| \le \rho \quad \forall (u \in \mathcal{U}, \ \xi \in \Xi). \tag{3}$$

*Then for all $1 \le p \le q$ and $\upsilon > 0$,*

$$\text{Risk}_p(B\widehat{x}_{\text{reg}}(y)|s, D, \upsilon, \epsilon) \le (4s)^{\frac{1}{p}}\frac{2\rho + s^{-1}\upsilon}{1 - 2\kappa}, \ 1 \le p \le q. \tag{4}$$

**Penalized $\ell_1$ recovery**   is

$$\widehat{x}_{\text{pen}}(y) \in \underset{z}{\text{Argmin}} \ \left\{ L_1(Bz) + 2s\|H^T(Az - y)\| \right\},$$

where $H \in \mathbb{R}^{m \times M}$, $\|\cdot\|$ and a positive integer $s$ are parameters of the construction. The accuracy of the penalized recovery is given by the following analogue of Theorem 2.1:

**Theorem 2.2** *Let $s$ be a positive integer, $q \in [1, \infty]$ and $\kappa \in (0, 1/2)$. Let also $\epsilon \in (0, 1)$. Assume that the parameters $H$, $\|\cdot\|$, $s$ of the penalized recovery and a $\rho \ge 0$ satisfy conditions **A**, **B** from Theorem 2.1. Then for all $1 \le p \le q$ and $\upsilon > 0$ we have*

$$\text{Risk}_p(B\widehat{x}_{\text{pen}}(y)|s, D, \upsilon, \epsilon) \le 2(2s)^{\frac{1}{p}}\frac{2\rho + s^{-1}\upsilon}{1 - 2\kappa}, \ 1 \le p \le q, \tag{5}$$

*cf. (4).*

# 3 Evaluating Condition $\mathbf{Q}_{s,\infty}(\kappa)$

The condition $\mathbf{Q}_{s,q}(\kappa)$, of Section 2.2, is closely related to known conditions, introduced to study the properties of recovery routines in the context of block-sparsity. Let us consider the representation structure with $B = I_n$. If the norm $\|\cdot\|$ in (2) is chosen to be the $\ell_\infty$-norm, we have the following obvious observation:

> (!) *Let $H$ satisfy $\mathbf{Q}_{s,q}(\kappa)$ and let $\widehat{\lambda}$ be the maximum of the Euclidean norms of columns in $H$. Then*
> $$\forall x \in \mathbb{R}^n : \ L_{s,q}(x) \leq \widehat{\lambda} s^{\frac{1}{q}} \|Ax\|_2 + \kappa s^{\frac{1}{q}-1} L_1(x).$$

Note that conditions of this kind with $\kappa < 1/2$ and $\|\cdot\|_r = \|\cdot\|_2$ play a crucial role in the performance analysis of group-Lasso and Dantzig Selector. For example, the error bounds for Lasso recovery, obtained in [18] rely upon the Restricted Eigenvalue assumption $\mathrm{RE}(s, \varkappa)$ which is as follows: there is $\varkappa > 0$ such that

$$L_2(x^s) \leq \frac{1}{\varkappa} \|Ax\|_2 \text{ whenever } 3L_1(x^s) \geq L_1(x - x^s).$$

Hence, $L_{s,1}(x) \leq \sqrt{s} L_{s,2}(x) \leq \frac{\sqrt{s}}{\varkappa} \|Ax\|_2$ whenever $4L_{s,1}(x) \geq L_1(x)$, so that

$$\forall x \in \mathbb{R}^n : L_{s,1}(x) \leq \frac{s^{1/2}}{\varkappa} \|Ax\|_2 + \frac{1}{4} L_1(x) \tag{6}$$

(observe that (6) is nothing but the "block version" of the Compatibility condition from [4]).

The bad news is that, in general, condition $\mathbf{Q}_{s,q}(\kappa)$, as well as RE and Compatibility conditions, cannot be verified. Specifically, given a sensing matrix $A$ and a representation structure $\mathcal{B}$, it seems to be difficult even to verify that a pair $(H, \|\cdot\|)$ satisfies condition $\mathbf{Q}_{s,q}(\kappa)$ associated with $A$, $B$, let alone to synthesize such $H$ which satisfies this condition and results in the best possible error bounds (4), (5) for the regular and the penalized $\ell_1$-recoveries. The good news is that when $\|\cdot\|_r$ is the uniform norm $\|\cdot\|_\infty$ and, in addition, $q = \infty$ the condition $\mathbf{Q}_{s,q}(\kappa)$ becomes "fully computationally tractable".[1] We intend to demonstrate also that this condition $\mathbf{Q}_{s,\infty}(\kappa)$ is in fact necessary for the bounds of the form (4), (5) to be valid when $p = \infty$.

## 3.1 Condition $\mathbf{Q}_{s,\infty}(\kappa)$, case $r = \infty$: tractability

Consider the case of the representation structure $\mathcal{B}_\infty = (B, n_1, ... n_K, \|\cdot\|_\infty)$. We have the following result.

**Proposition 3.1** *Let $\|\cdot\|_{(k)} = \|\cdot\|_\infty$ for all $k \leq K$, and let a positive integer $s$ and reals $\kappa > 0$, $\epsilon \in (0,1)$ be given.*

(i) *Assume that a triple $(H, \|\cdot\|, \rho)$, where $H \in \mathbb{R}^{M \times m}$, $\|\cdot\|$ is a norm on $\mathbb{R}^M$, and $\rho \geq 0$, is such that*

> (!) *$(H, \|\cdot\|)$ satisfies $\mathbf{Q}_{s,\infty}(\kappa)$ and the set $\Xi = \{\xi : \|H^T[u + D\xi]\| \leq \rho \ \forall u \in \mathcal{U}\}$ is such that $\mathrm{Prob}(\xi \in \Xi) \geq 1 - \epsilon$.*

*Then there exist $N = n_1 + ... + n_K$ vectors $h^1, ..., h^N$ in $\mathbb{R}^m$ and $N \times N$ block matrix $V = [V^{k\ell}]_{k,\ell=1}^K$ (the blocks $V^{k\ell}$ of $V$ are $n_k \times n_\ell$ matrices) such that*

> (a)    $B = VB + [h^1, ..., h^N]^T A$,
> (b)    $\|V^{k\ell}\|_{\infty,\infty} \leq s^{-1}\kappa \quad \forall k, \ell \leq K$
> *(here $\|V^{k\ell}\|_{\infty,\infty} = \max_{1 \leq j \leq n_\ell} \|\mathrm{Row}_j(V^{k\ell})\|_1$, $\mathrm{Row}_j(M)$ being the $j$-th row of $M$),*(7)
> (c)    $\mathrm{Prob}\left(\Xi^+ := \{\xi : \max_{u \in \mathcal{U}} u^T h^i + |(D\xi)^T h^i| \leq \rho, 1 \leq i \leq N\}\right) \geq 1 - \epsilon.$

(ii) *Whenever vectors $h^1, ..., h^N \in \mathbb{R}^m$ and a matrix $V = [V^{k\ell}]_{k,\ell=1}^K$ satisfy (7), the $m \times N$ matrix $\widehat{H} = [h^1, ..., h^N]$, the norm $\|\cdot\|_\infty$ on $\mathbb{R}^N$ and $\rho$ form a triple satisfying (!).*

**Discussion.** Let a sensing matrix $A \in \mathbb{R}^{m \times n}$ and a representation structure $\mathcal{B}_\infty$ be given, along with a positive integer $s$, an uncertainty set $\mathcal{U}$, and quantities $D$ and $\epsilon$. Recall that Theorems 2.1, 2.2 say that if a triple $(H, \|\cdot\|, \rho)$ is such that $(H, \|\cdot\|)$ satisfies $\mathbf{Q}_{s,\infty}(\kappa)$ with $\kappa < 1/2$ and $H, \rho$ are such that for the set

$$\Xi = \{\xi : \|H^T[u + D\xi]\| \le \rho \,\forall u \in \mathcal{U}\}$$

it holds $P(\Xi) \ge 1 - \epsilon$, then for all $\upsilon \ge 0$, for the regular $\ell_1$ recovery associated with $(H, \|\cdot\|, \rho)$ and for the penalized $\ell_1$ recovery associated with $(H, \|\cdot\|, s)$ one has:

$$\text{Risk}_p(B\widehat{x}|s, D, \upsilon, \epsilon) \le 2(2s)^{\frac{1}{p}} \frac{2\rho + s^{-1}\upsilon}{1 - 2\kappa}, \ 1 \le p \le \infty. \tag{8}$$

Proposition 3.1 states that when applying this result, we lose nothing by restricting ourselves with triples $H = [h^1, ..., h^N] \in \mathbb{R}^{m \times N}$, $N = n_1 + ... + n_K$, $\|\cdot\| = \|\cdot\|_\infty$ on $\mathbb{R}^N$, $\rho \ge 0$ which can be augmented by an appropriately chosen $N \times N$ matrix $V$ to satisfy relations (7). In the rest of this discussion, it is assumed that we are speaking about triples $(H, \|\cdot\|, \rho)$ satisfying the just defined restrictions.

Now, as far as bounds (8) are concerned, they are completely determined by two parameters — $\kappa$ (which should be $< 1/2$) and $\rho$; the smaller are these parameters, the better are the bounds. In what follows we address the issue of efficient synthesis of matrices $H$ with "as good as possible" values of $\kappa$ and $\rho$.

Observe, first, that $H = [h^1, ..., h^N]$ and $\kappa$ should admit an extension by a matrix $V$ to a solution of the system of constraints (7). Let $\mu_{\mathcal{U}}(h) = \max\limits_{u \in \mathcal{U}} u^T h$. Note that the restriction

$$\text{Prob}\left(\Xi^+ = \left\{\xi : \mu_{\mathcal{U}}(h^i) + |(D\xi)^T h^i| \le \rho, \ 1 \le i \le N\right\}\right) \ge 1 - \epsilon, \tag{9}$$

implies that

$$\rho \ge \max_{1 \le i \le N} \left[\mu_{\mathcal{U}}(h^i) + \text{erfinv}\left(\frac{\epsilon}{2}\right)\|D^T h^i\|_2\right],$$

where $\text{erfinv}(\cdot)$ is the inverse error function[2], and it is implied by

$$\rho \ge \left[\mu_{\mathcal{U}}(h^i) + \text{erfinv}\left(\frac{\epsilon}{2N}\right)\|D^T h^i\|_2\right], \ 1 \le i \le N. \tag{10}$$

Ignoring the "gap" between $\text{erfinv}(\epsilon/2)$ and $\text{erfinv}\left(\frac{\epsilon}{2N}\right)$, we can safely model the restriction (9) by the system of convex constraints (10). Thus, the set $G_s$ of admissible $\kappa, \rho$ can be safely approximated by the computationally tractable convex set

$$\begin{aligned}
G_s^* &= \Big\{(\kappa, \rho) : \ \exists H = [h^1, ..., h^N] \in \mathbb{R}^{m \times N}, \ V = [V^{k\ell} \in \mathbb{R}^{n_k \times n_\ell}]_{k,\ell=1}^K \text{ s.t.} \\
&\qquad B = BV + H^T A, \ \|V^{k\ell}\|_{\infty,\infty} \le \frac{\kappa}{s}, \ 1 \le k, \ell \le K \\
&\qquad \max_{u \in \mathcal{U}} u^T h^i + \text{erfinv}\left(\frac{\epsilon}{2N}\right)\|D^T h^i\|_2 \le \rho, \ 1 \le i \le N\Big\}
\end{aligned}$$

## 3.2 Condition $\mathbf{Q}_{s,\infty}(\kappa)$, case $r = \infty$: necessity

Let the representation structure $\mathcal{B}_\infty = (B, n_1, ..., n_K, \|\cdot\|_\infty)$ be fixed. From the above discussion we know that if, for some $\kappa < 1/2$ and $\rho > 0$, there exist $H = [h^1, ..., h^N] \in \mathbb{R}^{m \times N}$ and $V = [V^{k\ell} \in \mathbb{R}^{n_k \times n_\ell}]_{k,\ell=1}^K$ satisfying (7), then regular $\ell_1$-recovery with appropriate choice of parameters ensures that

$$\text{Risk}_\infty(B\widehat{x}_{\text{reg}}|s, D, \upsilon, \epsilon) \le \frac{2\rho + s^{-1}\upsilon}{1 - 2\kappa}. \tag{11}$$

We are about to demonstrate that this implication can be "nearly inverted:"

**Proposition 3.2** *Let a sensing matrix $A$, an uncertainty set $\mathcal{U}$, and reals $\kappa > 0$, $\epsilon \in (0,1)$ be given. Assume also that the observation error "is present," specifically, that for every $r > 0$, the set $\{u + De : u \in \mathcal{U}, \|e\|_2 \le r\}$ contains a neighborhood of the origin.*

*Given a positive integer $S$, assume that there exists a recovering routine $\widehat{x}$ satisfying an error bound of the form* (11), *namely,*

$$\forall (x \in \mathbb{R}^n, u \in \mathcal{U}) : \text{Prob}_\xi \{\|B[\widehat{x}(y) - x]\|_\infty \le \alpha + S^{-1}L_1(Bx - [Bx]^S)\} \ge 1 - \epsilon. \quad (12)$$

*for some $\alpha > 0$. Then there exist $H = [h^1, ..., h^N] \in \mathbb{R}^{m \times N}$ and $V = [V^{k\ell} \in \mathbb{R}^{n_k \times n_\ell}]_{k,\ell=1}^K$, satisfying*

$$(a) \qquad B = VB + H^T A,$$
$$(b) \qquad \|V^{k\ell}\|_{\infty,\infty} \le 2S^{-1} \ \ \forall k, \ell \le K,$$

*with*

$$(c) \qquad \rho := \max_{1 \le i \le N}\left[\max_{u \in \mathcal{U}} u^T h^i + \text{erfinv}(\frac{\epsilon}{2N})\|D^T h^i\|_2\right] \le 2\alpha \frac{\text{erfinv}\left(\frac{\epsilon}{2N}\right)}{\text{erfinv}\left(\frac{\epsilon}{2}\right)},$$

*and such that*

$$\text{Prob}\left(\Xi^+ := \{\xi : \max_{u \in \mathcal{U}} u^T h^i + |(D\xi)^T h^i| \le \rho, \, 1 \le i \le N\}\right) \ge 1 - \epsilon.$$

The latter exactly means that the exhibited $H$ satisfies the condition $\mathbf{Q}_{s,\infty}(\kappa)$ (see Proposition 3.1) for $s$ "nearly as large as $S$," namely, $s \le \frac{\kappa S}{2}$. Further, $H = [h^1, ..., h^k]$, $\rho$ satisfy conditions (10) (and thus – condition **B** of Theorem 2.1), with $\rho$ being "nearly $\alpha$", namely, $\rho \le 2\alpha\frac{\text{erfinv}(\frac{\epsilon}{2N})}{\text{erfinv}(\frac{\epsilon}{2})}$. As a consequence, under the premise of the proposition, we have for $s \le \frac{S}{8}$ (cf (11)):

$$\text{Risk}_\infty(B\widehat{x}_{\text{reg}}|s, D, \upsilon, \epsilon) \le 8\alpha\frac{\text{erfinv}(\frac{\epsilon}{2N})}{\text{erfinv}(\frac{\epsilon}{2})} + 2s^{-1}\upsilon.$$

### 3.3  Condition $\mathbf{Q}_{s,\infty}(\kappa)$, case $r = 2$: a verifiable sufficient condition

In this section we consider the case of the representation structure $\mathcal{B}_2 = (B, n_1, ..., n_K, \|\cdot\|_2)$. A verifiable sufficient condition for $Q_{s,\infty}(\kappa)$ is given by the following

**Proposition 3.3** *Let a sensing matrix $A$, a representation structure $\mathcal{B}_2$ be given. Let $N = n_1 + ... + n_K$, and let $N \times N$ matrix $V = [V^{k\ell}]_{k,\ell=1}^K$ ($V^{k\ell}$ are $n_k \times n_\ell$) and $m \times N$ matrix $H$ satisfy the relation*

$$B = VB + H^T A. \quad (13)$$

*Let*

$$\nu^*(V) = \max_{1 \le k, \ell \le K} \sigma_{\max}(V^{k\ell}), \quad (14)$$

*where $\sigma_{\max}$ stands for the maximal singular value. Then for all $s \le K$ we have:*

$$L_{s,\infty}(Bx) \le L_\infty(H^T Ax) + \nu^*(V)L_1(Bx), \ \ \forall x. \quad (15)$$

Suppose that the matrix $A$, the representation structure $\mathcal{B}_2$, the uncertainty set $\mathcal{U}$, and the parameters $D$, $\epsilon$ are given. Let us assume that the triple $H$, $\|\cdot\| = L_\infty(\cdot)$, and $\rho$ can be augmented by an appropriately chosen block $N \times N$ matrix $V$ to satisfy the system of convex constraints (13), (14). Our objective now is to synthesize the matrix $H = [H^k \in \mathbb{R}^{m \times n_k}]_{k=1}^K$ which satisfies the relationship (3) with "as good as possible" value of $\rho$.

Let us compute a bound on the probability of deviations of the variable $\|(H^k)^T D\xi\|_2$. Note that the distribution of $\|(H^k)^T D\xi\|_2^2$ coincides with that of the random variable $\zeta_k = \sum_{k=1}^{n_k} v_i[k]\eta_i^2$, where $\eta, ..., \eta_{n_k}$ are i.i.d $\mathcal{N}(0,1)$ and $v[k] = [\sigma_1^2[k], ..., \sigma_{n_k}^2[k]]$, $\sigma_i[k]$ being the principal singular values of $(H^k)^T D$. To bound the deviation probabilities for $\zeta_k$ we use the bound of [6] for the deviation of the weighted $\chi^2$:

$$\text{Prob}\left\{\sum_{i=1}^{n_k} v_i[k]\eta_i^2 \ge \|v[k]\|_1 + \sqrt{2}\|v[k]\|_2\tau\right\} \le 2\exp\left(-\frac{\tau^2}{4\|v[k]\|_2^2 + 4\tau\|v[k]\|_\infty}\right).$$

When substituting $\|v[k]\|_\infty = \sigma_{\max}^2[k]$, $\|v[k]\|_2 \leq \sigma_{\max}^2[k]\sqrt{n_k}$, and $\|v[k]\|_1 = \|\sigma[k]\|_2^2$, where $\sigma_{\max}[k]$ is the maximal singular value and $\|\sigma[k]\|_2$ is the Frobenius norm of $H^T D$, after a simple algebra we come to

$$\mathrm{Prob}\left\{\|(D\xi)^T H[k]\|_2 \geq \|\sigma[k]\|_2 + \sigma_{\max}[k]\sqrt{4\ln(2K\epsilon^{-1}) + 2\sqrt{n_k \ln(2K\epsilon^{-1})}}\right\} \leq \frac{\epsilon}{K}.$$

Let $\mu_{\mathcal{U}}(H_k) = \max_{u \in \mathcal{U}} \|u^T H[k]\|_2$. Then the chance constraint

$$\mathrm{Prob}\left\{\xi : \mu_{\mathcal{U}}(H^k) + \|(D\xi)^T H[k]\|_2 \leq \rho,\ 1 \leq k \leq K\right\} \geq 1 - \epsilon,$$

is satisfied for

$$\rho \geq \max_k \left[\mu_{\mathcal{U}}(H^k) + \|D^T H[k]\|_F + \sigma_{\max}(D^T H[k])\sqrt{4\ln(2K\epsilon^{-1}) + 2\sqrt{n_k \ln(2K\epsilon^{-1})}}\right]$$

(here $\|\cdot\|_F$ stands for the Frobenius norm). In particular, in the case $\mathcal{U} = \{0\}$ (there is no nuisance), the set $G_s$ of admissible $\kappa, \rho$ can be safely approximated by the computationally tractable convex set

$$
\begin{aligned}
G_s^* \quad = \quad &\Big\{(\kappa, \rho) : \quad \exists H = [H^k \in \mathbb{R}^{m \times n_k}]_{k=1}^K, \quad V = [V^{k\ell} \in \mathbb{R}^{n_k \times n_\ell}]_{k,\ell=1}^K \\
&\qquad B = BV + H^T A, \quad \sigma_{\max}(V^{k\ell}) \leq \frac{\kappa}{s},\ 1 \leq k, \ell \leq K \\
&\rho \geq \|D^T H[k]\|_F + \sigma_{\max}(D^T H[k])\sqrt{4\ln(2K\epsilon^{-1}) + 2\sqrt{n_k \ln(2K\epsilon^{-1})}},\ 1 \leq k \leq K\Big\}.
\end{aligned}
$$

We have mentioned in Introduction that, to the best of our knowledge, the only previously proposed *verifiable* sufficient condition for the validity of block $\ell_1$ recovery is the Mutual Block Incoherence condition [9]. We aim now to demonstrate that this condition is covered by Proposition 3.3.

The Mutual Block Incoherence condition deals with the case where $B = I$ and all block norms are $\|\cdot\|_2$-norms. Let the sensing matrix $A$ in question be partitioned as $A = [A[1], ..., A[K]]$, where $A[k]$, $k = 1, ..., K$, has $n_k$ columns. Let us define the *mutual block-incoherence* $\mu$ of $A$ w.r.t. the representation structure in question as follows:

$$\mu = \max_{\substack{1 \leq k, \ell \leq K, \\ k \neq \ell}} \sigma_{\max}\left(C_k^{-1} A^T[k]A[\ell]\right), \quad [C_k := A^T[k]A[k]] \tag{16}$$

provided that all matrices $C_k$, $1 \leq k \leq K$, are nonsingular, otherwise $\mu = \infty$. Note that in the case of the standard representation structure, the just defined quantity is nothing but the standard mutual incoherence known from the Compressed Sensing literature.

We have the following observation.

**Proposition 3.4** *Given $m \times n$ sensing matrix $A$ and a representation structure $\mathcal{B}_2$ with $B = I$, $1 \leq k \leq K$, let $A = [A[1], ..., A[K]]$ be the corresponding partition of $A$.*
*Let $\mu$ be the mutual block-incoherence of $A$ with respect to $\mathcal{B}_2$. Assuming $\mu < \infty$, we set*

$$H = \frac{1}{1+\mu}[A[1]C_1^{-1}, A[2]C_2^{-1}, ..., A[K]C_K^{-1}], \quad C_k = A^T[k]A[k]. \tag{17}$$

*Then the contrast matrix $H$ along with the matrix $V = I - H^T A$ satisfies condition (13) (where $B = I$) and condition (14) with $\nu^*(V) \leq \frac{\mu s}{1+\mu}$. As a result, applying Proposition 3.3, we conclude that whenever*

$$s < \frac{1+\mu}{2\mu}, \tag{18}$$

*the pair $(H, L_\infty(\cdot))$ satisfies $\mathbf{Q}_{s,\infty}(\kappa)$ with $\kappa = \frac{\mu s}{1+\mu} < 1/2$.*

Note that Proposition 3.4 essentially covers the results of [9] where the authors prove, under a condition which is marginally stronger than that of (18), that an appropriate version of block-$\ell_1$ recovery allows to recover exactly every block-sparse signal from the noiseless observation $y = Ax$.

## Footnotes

[1]Recall that by Observation 2.1, $q = \infty$ corresponds to the strongest among the conditions $\mathbf{Q}_{s,q}(\kappa)$ associated with $A$ and a given representation structure $\mathcal{B}$ and ensures the validity of the bounds (4) and (5) in the largest possible range, $1 \leq p \leq \infty$, of values of $p$.

[2]i.e., $u = \text{erfinv}(\delta)$ means that $\frac{1}{\sqrt{2\pi}} \int_u^\infty e^{-t^2/2} dt = \delta$.

# References

[1] F. Bach. Consistency of the group lasso and multiple kernel learning. *J. Mach. Learn. Res.*, 9:1179–1225, 2008.

[2] Z. Ben-Haim and Y. C. Eldar. Near-oracle performance of greedy block-sparse estimation techniques from noisy measurements. Technical report, 2010. http://arxiv.org/abs/1009.0906.

[3] P. J. Bickel, Y. Ritov, and A. B. Tsybakov. Simultaneous analysis of Lasso and Dantzig selector. *Annals of Stat.*, 37(4):1705–1732, 2008.

[4] P. Bühlmann and S. van de Geer. On the conditions used to prove oracle results for the Lasso. *Electron. J. Statist.*, 3:1360–1392, 2009.

[5] E. J. Candès and T. Tao. Decoding by linear programming. *IEEE Trans. Inf. Theory*, 51:4203–4215, 2006.

[6] L. Cavalier, G. K. Golubev, D. Picard, and A. B. Tsybakov. Oracle inequalities for inverse problems. *Ann. Statist.*, 30(3):843–874, 2002.

[7] A. Chambolle. An algorithm for total variation minimization and applications. *Journal of Mathematical Imaging and Vision*, 20(1-2):89–97, 2004.

[8] C. Chesneau and M. Hebiri. Some theoretical results on the grouped variables Lasso. *Mathematical Methods of Statistics*, 27(4):317–326, 2008.

[9] Y. C. Eldar, P. Kuppinger, and H. Bölcskei. Block-sparse signals: Uncertainty relations and efficient recovery. *IEEE Trans. on Signal Processing*, 58(6):3042–3054, 2010.

[10] Y. C. Eldar and M. Mishali. Robust recovery of signals from a structured union of subspaces. *IEEE Trans. Inf. Theory*, 55(11):5302–5316, 2009.

[11] J. Huang and T. Zhang. The benefit of group sparsity. *Annals of Stat.*, 38(4):1978–2004, 2010.

[12] G. M. James, P. Radchenko, and J. Lv. Dasso: connections between the Dantzig selector and Lasso. *J. Roy. Statist. Soc. Ser. B*, 71(1):127–142, 2009.

[13] A. B. Juditsky, F. Kılınç-Karzan, and A. S. Nemirovski. Verifiable conditions of $\ell_1$ recovery for sparse signals with sign restrictions. *Math. Progr.*, 127(1):89–122, 2010. http://www.optimization-online.org/DB_HTML/2009/03/2272.html.

[14] A. B. Juditsky and A. S. Nemirovski. Accuracy guarantees for $\ell_1$-recovery. Technical report, 2010. http://www.optimization-online.org/DB_HTML/2010/10/2778.html.

[15] A. B. Juditsky and A. S. Nemirovski. On verifiable sufficient conditions for sparse signal recovery via $\ell_1$ minimization. *Math. Progr.*, 127(1):57–88, 2010. Special issue on machine learning.

[16] H. Liu and J. Zhang. Estimation consistency of the group Lasso and its applications. *Journal of Machine Learning Research - Proceedings Track*, 5:376–383, 2009.

[17] H. Liu, J. Zhang, X. Jiang, and J. Liu. The group Dantzig selector. *Journal of Machine Learning Research - Proceedings Track*, 9:461–468, 2010.

[18] K. Lounici, M. Pontil, A. Tsybakov, and S. van de Geer. Oracle inequalities and optimal inference under group sparsity. Technical report, 2010. http://arxiv.org/pdf/1007.1771.

[19] Y. Nardi and A. Rinaldo. On the asymptotic properties of the group Lasso estimator for linear models. *Electron. J. Statist.*, 2:605–633, 2008.

[20] G. Obozinski, M. J. Wainwright, and M. I. Jordan. Support union recovery in high-dimensional multivariate regression. *Annals of Stat.*, 39(1):1–47, 2011.

[21] L. Rudin, S. Osher, and E. Fatemi. Nonlinear total variation based noise removal algorithms. *Physica D*, 60(1-4):259–268, 1992.

[22] M. Stojnic, F. Parvaresh, and B. Hassibi. On the reconstruction of block-sparse signals with an optimal number of measurements. *IEEE Trans. on Signal Processing*, 57(8):3075–3085, 2009.

[23] M. Yuan and Y. Lin. Model selection and estimation in regression with grouped variables. *J. Roy. Stat. Soc. Ser. B*, 68(1):49–67, 2006.

